# Proximity graphs for clustering and manifold learning

**Miguel Á. Carreira-Perpiñán**          **Richard S. Zemel**
Dept. of Computer Science, University of Toronto
6 King's College Road. Toronto, ON   M5S 3H5, Canada
Email: {miguel,zemel}@cs.toronto.edu

## Abstract

Many machine learning algorithms for clustering or dimensionality reduction take as input a cloud of points in Euclidean space, and construct a graph with the input data points as vertices. This graph is then partitioned (clustering) or used to redefine metric information (dimensionality reduction). There has been much recent work on new methods for graph-based clustering and dimensionality reduction, but not much on constructing the graph itself. Graphs typically used include the fully-connected graph, a local fixed-grid graph (for image segmentation) or a nearest-neighbor graph. We suggest that the graph should adapt locally to the structure of the data. This can be achieved by a graph ensemble that combines multiple minimum spanning trees, each fit to a perturbed version of the data set. We show that such a graph ensemble usually produces a better representation of the data manifold than standard methods; and that it provides robustness to a subsequent clustering or dimensionality reduction algorithm based on the graph.

## 1   Introduction

Graph-based algorithms have long been popular, and have received even more attention recently, for two of the fundamental problems in machine learning: clustering [1–4] and manifold learning [5–8]. Relatively little attention has been paid to the properties and construction methods for the graphs that these algorithms depend on.

A starting point for this study is the question of what constitutes a good graph. In the applications considered here, the graphs are an intermediate form of representation, and therefore their utility to some extent depends on the algorithms that they will ultimately be used for. However, in the case of both clustering and manifold learning, the data points are assumed to lie on some small number of manifolds. Intuitively, the graph should represent these underlying manifolds well: it should avoid *shortcuts* that travel outside a manifold, avoid *gaps* that erroneously disconnect regions of a manifold, and be dense within the manifold and clusters. Also, while the algorithms differ with respect to connectedness, in that clustering wants the graph to be disconnected, while for manifold learning the graph should be connected, they both want at least the inside of the clusters, or dense areas of the manifold, to be enhanced relative to the between-cluster, or sparse manifold connections.

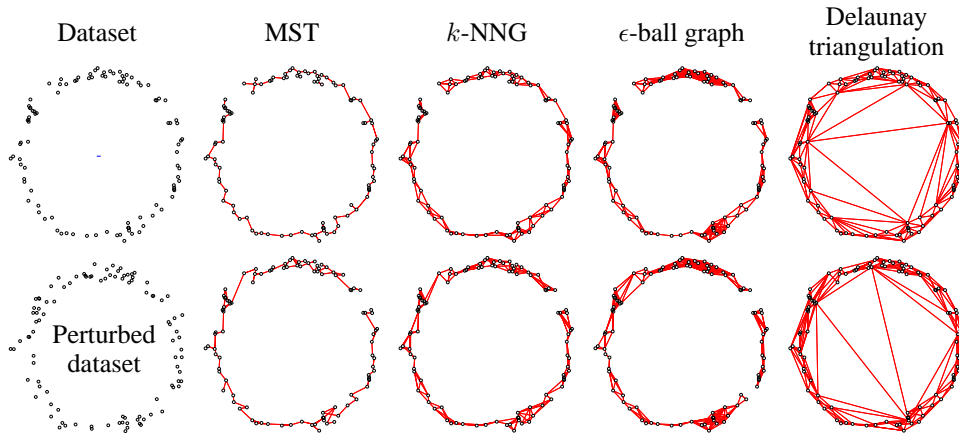

| Dataset | MST | $k$-NNG | $\epsilon$-ball graph | Delaunay triangulation |

Perturbed dataset

Figure 1: Sensitivity to noise of proximity graphs. *Top row*: several proximity graphs constructed on a noisy sample of points lying on a circle. *Bottom row*: the same graphs constructed on a different sample; specifically, we added to each point Gaussian noise of standard deviation equal to the length of the small segment shown in the centre of the dataset (top row), built the graph, and drew it on the original dataset. This small perturbation can result in large changes in the graphs, such as disconnections, shortcuts or changes in connection density.

Many methods employ simple graph constructions. A fully-connected graph is used for example in spectral clustering and multidimensional scaling, while a *fixed grid*, with each point connecting to some small fixed number of neighbors in a pre-defined grid of locations, is generally used in image segmentation. An $\epsilon$-*ball*, in which each point connects to points within some distance $\epsilon$, and $k$-nearest neighbors ($k$-NNG) are generalization of these approaches, as they take into account distance in some features associated with each point instead of simply the grid locations. The $\epsilon$-ball or $k$-NNG provide an improvement over the fully-connected graph or fixed grid (clustering: [3, 9]; manifold learning: [5, 7]). These traditional methods contain parameters ($\epsilon$, $k$) that strongly depend on the data; they generally require careful, costly tuning, as typically graphs must be constructed for a range of parameter values, the clustering or dimensionality-reduction algorithm run on each, and then performance curves compared to determine the best settings. Figure 1 shows that these methods are quite sensitive to sparsity and noise in the data points, and that the parameters should ideally vary within the data set. It also shows that other traditional graphs (e.g. the Delaunay triangulation) are not good for manifolds, since they connect points nonlocally.

In this paper we propose a different method of graph construction, one based on *minimum spanning trees* (MSTs). Our method involves an ensemble of trees, each built on a perturbed version of the data. We first discuss the motivation for this new type of graph, and then examine its robustness properties, and its utility to both subsequent clustering or dimensionality reduction methods.

## 2 Two new types of proximity graphs

A minimum spanning tree is a tree subgraph that contains all the vertices and has a minimum sum of edge weights. As a skeleton of a data set, the MST has some good properties: it tends to avoid shortcuts between branches (typically caused by long edges, which are contrary to the shortest-length criterion) and it gives a connected graph (usually a problem for other methods with often-occurring random small groupings of points). In fact, the

MST was an early approach to clustering [10]. However, the MST is too sparse (having only $N-1$ edges for an $N$-point data set, and no cycles) and is sensitive to noise. One way to flesh it out and attain robustness to noise is to form an *MST ensemble* that combines multiple MSTs; we give two different algorithms for this.

## 2.1 Perturbed MSTs (PMSTs)

Perturbed MSTs combine a number of MSTs, each fit to a perturbed version of the data set. The perturbation is done through a *local noise model* that we estimate separately for each data point based on its environment: point $\mathbf{x}_i$ is perturbed by adding to it zero-mean uniform noise of standard deviation $s_i = r d_i$ where $d_i$ is the average distance to the $k$ nearest neighbors of $\mathbf{x}_i$, and $r \in [0, 1]$. In this paper we use $k = 5$ throughout and study the effect of $r$. The locality of the noise model allows points to move more or less depending on the local data structure around them and to connect to different numbers of neighbors at different distances—in effect we achieve a variable $k$ and $\epsilon$.

To build the PMST ensemble, we generate $T > 1$ perturbed copies of the entire data set according to the local noise model and fit an MST to each. The PMST ensemble assigns a weight $e_{ij} \in [0, 1]$ to the edge between points $\mathbf{x}_i$ and $\mathbf{x}_j$ equal to the average number of times that edge appears on the trees. For $T = 1$ this gives the MST of the unperturbed data set; for $T \to \infty$ it gives a stochastic graph where $e_{ij}$ is the probability (in the Laplace sense) of that edge under the noise model. The PMST ensemble contains at most $T(N-1)$ edges (usually much less). Although the algorithm is randomized, the PMST ensemble for large $T$ is essentially deterministic, and insensitive to noise by construction. In practice a small $T$ is enough; we use $T = 20$ in the experiments.

## 2.2 Disjoint MSTs (DMSTs)

Here we build a graph that is a deterministic collection of $t$ MSTs that satisfies the property that the $n$th tree (for $n = 1, \dots, t$) is the MST of the data subject to not using any edge already in the previous $1, \dots, t-1$ trees. One possible construction algorithm is an extension of Kruskal's algorithm for the MST where we pick edges without replacement and restart for every new tree. Specifically, we sort the list of $\frac{N(N-1)}{2}$ edges $e_{ij}$ by increasing distance $d_{ij}$ and visit each available edge in turn, removing it if it merges two clusters (or equivalently does not create a cycle); whenever we have removed $N-1$ edges, we go back to the start of the list. We repeat the procedure $t$ times in total.

The DMST ensemble consists of the union of all removed edges and contains $t(N-1)$ edges each of weight 1. The $t$ parameter controls the overall density of the graph, which is always connected; unlike $\epsilon$ or $k$ (for the $\epsilon$-ball or $k$-NNG), $t$ is not a parameter that depends locally on the data, and again points may connect to different numbers of neighbors at different distances. We obtain the original MST for $t = 1$; values $t = 2$–4 (and usually quite larger) work very well in practice. $t$ need not be integer, i.e., we can fix the total number of edges instead. In any case we should use $t \ll \frac{N}{2}$.

## 2.3 Computational complexity

For a data set with $N$ points, the computational complexity is approximately $\mathcal{O}(TN^2 \log N)$ (PMSTs) or $\mathcal{O}(N^2(\log N + t))$ (DMSTs). In both cases the resulting graphs are sparse (number of edges is linear on number of points $N$). If imposing an a priori sparse structure (e.g. an 8-connected grid in image segmentation) the edge list is much shorter, so the graph construction is faster. For the perturbed MST ensemble, the perturbation of the data set results in a partially disordered edge list, which one should be able to sort efficiently. The bottleneck in the graph construction itself is the computation of

pairwise distances, or equivalently of nearest neighbors, of a set of $N$ points (which affects the $\epsilon$-ball and $k$-NNG graphs too): in 2D this is $\mathcal{O}(N \log N)$ thanks to properties of planar geometry, but in higher dimensions the complexity quickly approaches $\mathcal{O}(N^2)$.

Overall, the real computational bottleneck is the graph postprocessing, typically $\mathcal{O}(N^3)$ in spectral methods (for clustering or manifold learning). This can be sped up to $\mathcal{O}(cN^2)$ by using sparsity (limiting a priori the edges allowed, thus approximating the true solution) but then the graph construction is likewise sped up. Thus, even if our graphs are slightly more costly to construct than the $\epsilon$-ball or $k$-NNG, the computational savings are very large if we avoid having to run the spectral technique multiple times in search for a good $\epsilon$ or $k$.

## 3 Experiments

We present two sets of experiments on the application of the graphs to clustering and manifold learning, respectively.

### 3.1 Clustering

In affinity-based clustering, our data is an $N \times N$ affinity matrix $\mathbf{W}$ that defines a graph (where nonzeros define edge weights) and we seek a partition of the graph that optimizes a cost function, such as mincut [1] or normalized cut [2]. Typically the affinities are $w_{ij} = \exp(-\frac{1}{2}(d_{ij}/\sigma)^2)$ (where $d_{ij}$ is the problem-dependent distance between points $\mathbf{x}_i$ and $\mathbf{x}_j$) and depend on a scale parameter $\sigma \in (0, \infty)$. This graph partitioning problem is generally NP-complete, so approximations are necessary, such as spectral clustering algorithms [2]. In spectral clustering we seek to cluster in the leading eigenvectors of the normalized affinity matrix $\mathbf{N} = \mathbf{D}^{-\frac{1}{2}}\mathbf{W}\mathbf{D}^{-\frac{1}{2}}$ (where $\mathbf{D} = \mathrm{diag}\left(\sum_i w_{ij}\right)$, and discarding a constant eigenvector associated with eigenvalue 1). Spectral clustering succeeds only for a range of values of $\sigma$ where $\mathbf{N}$ displays the natural cluster structure of the data; if $\sigma$ is too small $\mathbf{W}$ is approximately diagonal and if $\sigma$ is too large $\mathbf{W}$ is approximately a matrix of ones. It is thus crucial to determine a good $\sigma$, which requires computing clusterings over a range of $\sigma$ values—an expensive computation since each eigenvector computation is $\mathcal{O}(N^3)$ (or $\mathcal{O}(cN^2)$ under sparsity conditions).

Fig. 2 shows segmentation results for a grayscale image from [11] where the objective is to segment the occluder from the underlying background, a hard task given the intensity gradients. We use a standard weighted Euclidean distance on the data points (pixels) $\mathbf{x} =$ (pixel location, intensity). One method uses the 8-connected grid (where each pixel is connected to its 8 neighboring pixels). The other method uses the PMST or DMST ensemble (constrained to contain only edges in the 8-connected grid) under different values of the $r, t$ parameters; the graph has between 44% and 98% the number of edges in the 8-grid, depending on the parameter value. We define the affinity matrix as $w_{ij} = e_{ij} \exp(-\frac{1}{2}(d_{ij}/\sigma)^2)$ (where $e_{ij} \in [0, 1]$ are the edge values). In both methods we apply the spectral clustering algorithm of [2]. The plot shows the clustering error (mismatched occluder area) for a range of scales. The 8-connected grid succeeds in segmenting the occluder for $\sigma \in [0.2, 1]$ approximately, while the MST ensembles (for all parameter values tested) succeed for a wider range—up to $\sigma = \infty$ in many cases. The reason for this success even for such high $\sigma$ is that the graph lacks many edges around the occluder, so those affinities are zero no matter how high the scale is. In other words, for clustering, our graphs enhance the inside of clusters with respect to the bridges between clusters, and so ease the graph partitioning.

### 3.2 Manifold learning

For dimensionality reduction, we concentrate on applying Isomap [5], a popular and powerful algorithm. We first estimate the geodesic distances (i.e., along the manifold) $\hat{g}_{ij}$ be-

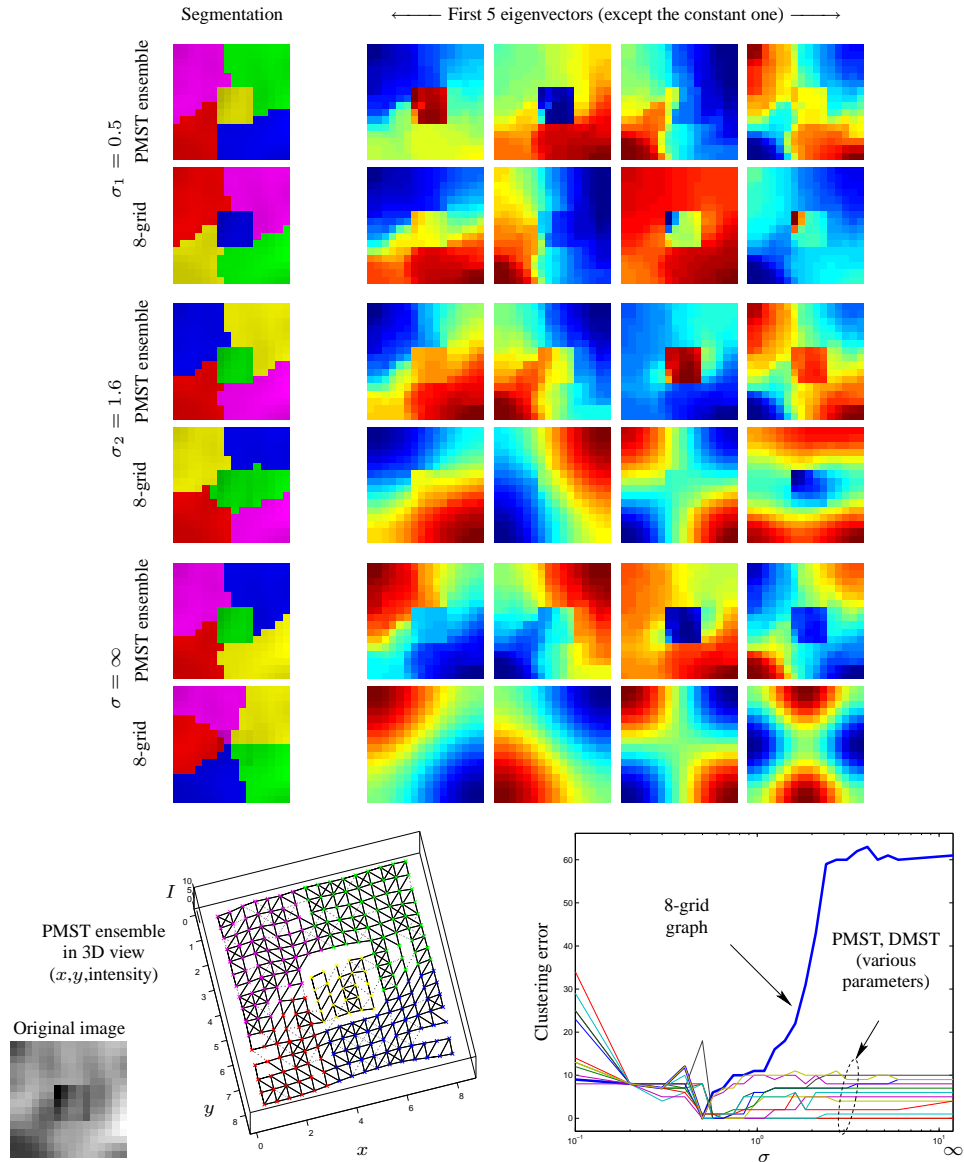

Figure 2: Using a proximity graph increases the scale range over which good segmentations are possible. We consider segmenting the greyscale image at the bottom (an occluder over a background) with spectral clustering, asking for $K = 5$ clusters. The color diagrams represent the segmentation (column 1) and first 5 eigenvectors of the affinity matrix (except the constant eigenvector, columns 2–4) obtained with spectral clustering, using a PMST ensemble with $r = 0.4$ (upper row) or an 8-connectivity graph (lower row), for 3 different scales: $\sigma_1 = 0.5$, $\sigma_2 = 1.6$ and $\sigma = \infty$. The PMST ensemble succeeds at all scales (note how several eigenvectors are constant over the occluder), while the 8-connectivity graph progressively deteriorates as $\sigma$ increases to give a partition of equal-sized clusters for large scale. In the bottom part of the figure we show: the PMST ensemble graph in 3D space; the clustering error vs $\sigma$ (where the right end is $\sigma = \infty$) for the 8-connectivity graph (thick blue line) and for various other PMST and DMST ensembles under various parameters (thin lines). The PMST and DMST ensembles robustly (for many settings of their parameters) give an almost perfect segmentation over a large range of scales.

tween pairs of points in the data set as the shortest-path lengths in a graph learned from the data. Then we apply multidimensional scaling to these distances to obtain a collection of low-dimensional points $\{\mathbf{y}_i\}_{i=1}^N$ that optimally preserves the estimated geodesic distances.

In fig. 3 we show the results of applying Isomap using different graphs to two data sets (ellipse and Swiss roll) for which we know the true geodesic distances $g_{ij}$. In a real application, since the true geodesic distances are unknown, error and variance cannot be computed; an *estimated residual variance* has been proposed [5] to determine the optimal graph parameter. For the perturbed MST ensemble, we binarize the edge values by making 1 any $e_{ij} > 0$. (It is often possible to refine the graph by zeroing edges with small $e_{ij}$, since this removes shortcuts that may have arisen by chance, particularly if $T$ is large; but it is difficult to estimate the right threshold reliably.) The plots show 3 curves as a function of the graph parameter: the average error $E$ in the geodesic distances; Isomap's estimated residual variance $\hat{V}$; and the true residual variance $V$. From the plots we can see that $\hat{V}$ correlates well with $V$ (though it underestimates it) and also with $E$ for the Swiss roll, but not for the ellipse; this can make the optimal graph parameter difficult to determine in a real application. Given this, the fact that our graphs work well over a larger region of their parameter space than the $\epsilon$-ball or $k$-NNG graphs makes them particularly attractive.

The plots for the Swiss roll show that, while for the low noise case the $\epsilon$-ball or $k$-NNG graphs work well over a reasonable region of their parameter space, for the high noise case this region decreases a lot, almost vanishing for the $\epsilon$-ball. This is because for low values of the parameter the graph is disconnected, while for high values it has multiple shortcuts; the difficulty of the task is compounded by the small number of points used, $N = 500$ (an unavoidable fact in high dimensions). However, for the PMSTs the region remains quite wide and for the DMSTs the approximate region $t \in [2, 8]$ gives good results. For very low $r$ or $t = 1$ the graph is the single MST, thus the large errors.

It is also important to realize that the range of the $r$ parameter of the PMST ensemble does not depend on the data, while the range for $\epsilon$ and $k$ does. The range of the $t$ parameter of the DMST ensemble does depend on the data, but we have found empirically that $t = 2$–$4$ gives very good results with all data sets we have tried.

## 4 Discussion

One main contribution of this paper is to highlight the relatively understudied problem of converting a data set into a graph, which forms an intermediate representation for many clustering and manifold learning algorithms. A second contribution is novel construction algorithms, which are: easy to implement, not expensive to compute, robust across many noise levels and parameter settings, and useful for clustering and manifold learning. In general, a careful selection of the graph construction algorithm makes the results of these machine learning methods robust, and avoids or limits the required parameter search. Finally, the combination of many graphs, formed from perturbed versions of the data, into an ensemble of graphs, is a novel approach to the construction problem.

Our idea of MST ensembles is an extension to graphs of the well-known technique of combining predictors by averaging (regression) or voting (classification), as is the regularizing effect of training with noise [12]. An ensemble of predictors improves the generalization to unseen data if the individual predictors are independent from each other and disagree with each other, and can be explained by the bias-variance tradeoff. Unlike regression or classification, unsupervised graph learning lacks at present an error function, so it seems difficult to apply the bias-variance framework here. However, we have conducted a wide range of empirical tests to understand the properties of the ensemble MSTs, and to compare them to the other graph construction methods, in terms of the error in the geodesic distances (if known a priori). In summary, we have found that the variance of the error for the geodesic

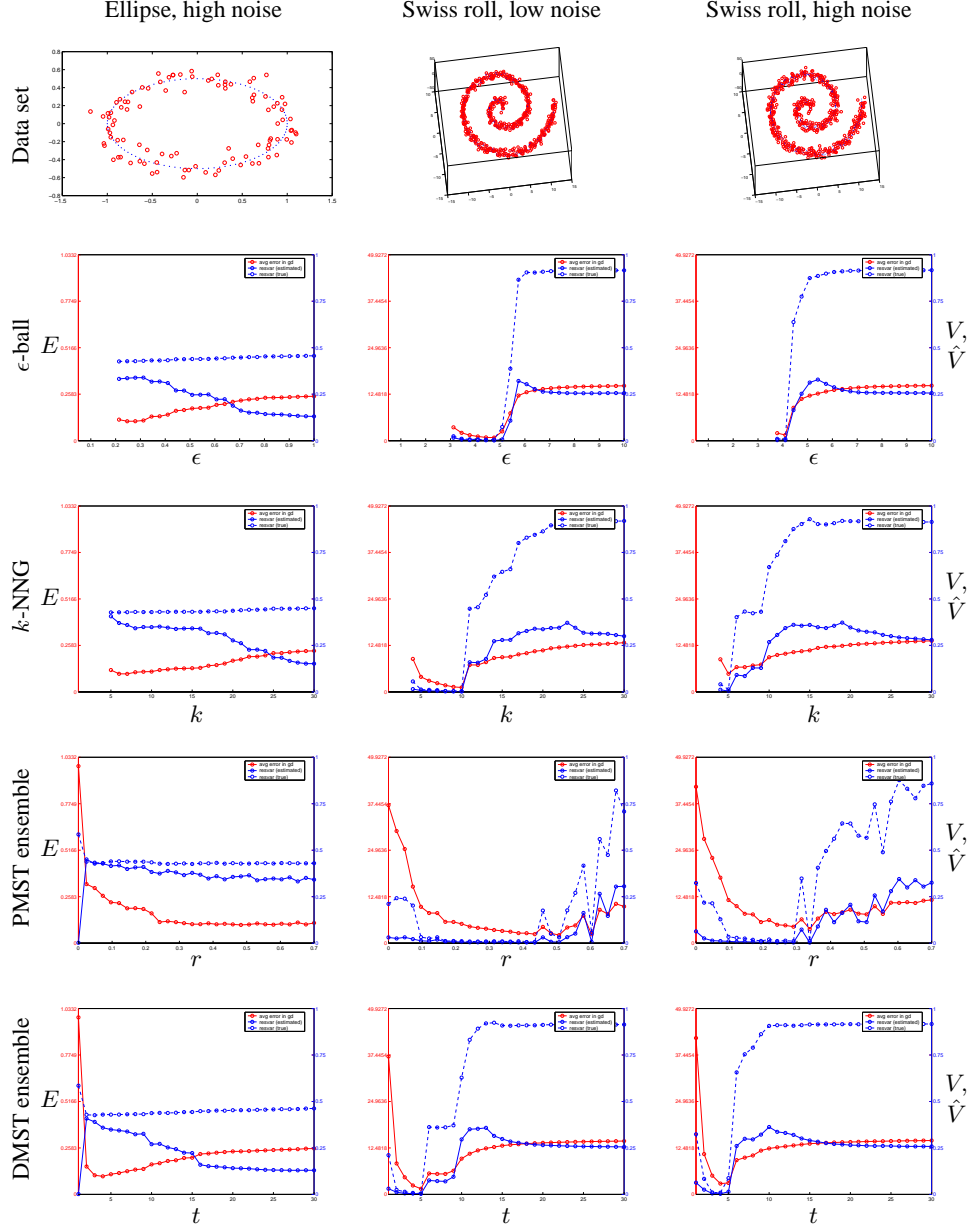

Figure 3: Performance of Isomap with different graphs in 3 data sets: ellipse with $N = 100$ points, high noise; Swiss roll with $N = 500$ points, low and high noise (where high noise means Gaussian with standard deviation equal to 9% of the separation between branches). All plots show on the X axis the graph parameter ($\epsilon$, $k$, $r$ or $t$); on the left Y axis the average error in the geodesic distances (red curve, $E = \frac{1}{N^2} \sum_{i,j=1}^{N} |\tilde{g}_{ij} - g_{ij}|$); and on the right Y axis Isomap's estimated residual variance (solid blue curve, $\hat{V} = 1 - R^2(\hat{\mathbf{G}}, \mathbf{D}_{\mathcal{Y}})$) and true residual variance (dashed blue curve, $V = 1 - R^2(\mathbf{G}, \mathbf{D}_{\mathcal{Y}})$), where $\hat{\mathbf{G}}$ and $\mathbf{G}$ are the matrices of estimated and true geodesic distances, respectively, $\mathbf{D}_{\mathcal{Y}}$ is the matrix of Euclidean distances in the low-dimensional embedding, and $R(\mathbf{A}, \mathbf{B})$ is the standard linear correlation coefficient, taken over all entries of matrices $\mathbf{A}$ and $\mathbf{B}$. Where the curves for $\epsilon$-ball and $k$-NNG are missing, the graph was disconnected.

distances decreases for the ensemble when the individual graphs are sparse (e.g. MSTs as used here, or $\epsilon$-ball and $k$-NNG with low $\epsilon$ or $k$); but not necessarily when the graphs are not sparse.

The typical cut [9, 13] is a clustering criterion that is based on the probability $p_{ij}$ that points $\mathbf{x}_i$ and $\mathbf{x}_j$ are in the same cluster over all possible partitions (under the Boltzmann distribution for the mincut cost function). The $p_{ij}$ need to be estimated: [9] use Swendsen-Wang sampling, while [13] use randomized trees sampling. However, these trees are not used to define a proximity graph, unlike in our work.

An important direction for future work concerns the noise model for PMSTs. The model we propose is isotropic, in that every direction of perturbation is equally likely. A better way is to perturb points more strongly in directions likely to lie within the manifold and less strongly in directions away from the manifold, using a method such as $k$ nearest neighbors to estimate appropriate directions. Preliminary experiments with such a manifold-aligned model are very promising, particularly when the data is very noisy or its distribution on the manifold is not uniform. The noise model can also be extended to deal with non-Euclidean data by directly perturbing the similarities.

### Acknowledgements

Funding provided by a CIHR New Emerging Teams grant.

## References

[1] Zhenyu Wu and Richard Leahy. An optimal graph theoretic approach to data clustering: Theory and its application to image segmentation. *IEEE Trans. on Pattern Anal. and Machine Intel.*, 15(11):1101–1113, November 1993.

[2] Jianbo Shi and Jitendra Malik. Normalized cuts and image segmentation. *IEEE Trans. on Pattern Anal. and Machine Intel.*, 22(8):888–905, August 2000.

[3] Pedro F. Felzenszwalb and Daniel P. Huttenlocher. Efficient graph-based image segmentation. *Int. J. Computer Vision*, 59(2):167–181, September 2004.

[4] Romer Rosales, Kannan Achan, and Brendan Frey. Learning to cluster using local neighborhood structure. In *ICML*, 2004.

[5] Joshua B. Tenenbaum, Vin de Silva, and John C. Langford. A global geometric framework for nonlinear dimensionality reduction. *Science*, 290(5500):2319–2323, December 22 2000.

[6] Sam T. Roweis and Lawrence K. Saul. Nonlinear dimensionality reduction by locally linear embedding. *Science*, 290(5500):2323–2326, December 22 2000.

[7] Mikhail Belkin and Partha Niyogi. Laplacian eigenmaps for dimensionality reduction and data representation. *Neural Computation*, 15(6):1373–1396, June 2003.

[8] Kilian Q. Weinberger and Lawrence K. Saul. Unsupervised learning of image manifolds by semidefinite programming. In *CVPR*, 2004.

[9] Marcelo Blatt, Shai Wiseman, and Eytan Domany. Data clustering using a model granular magnet. *Neural Computation*, 9(8):1805–1842, November 1997.

[10] C. T. Zahn. Graph-theoretical methods for detecting and describing gestalt clusters. *IEEE Trans. Computers*, C–20(1):68–86, April 1971.

[11] Chakra Chennubhotla and Allan Jepson. EigenCuts: Half-lives of EigenFlows for spectral clustering. In *NIPS*, 2003.

[12] Christopher M. Bishop. *Neural Networks for Pattern Recognition*. Oxford University Press, New York, Oxford, 1995.

[13] Yoram Gdalyahu, Daphna Weinshall, and Michael Werman. Self organization in vision: Stochastic clustering for image segmentation, perceptual grouping, and image database organization. *IEEE Trans. on Pattern Anal. and Machine Intel.*, 23(10):1053–1074, October 2001.
